# Probabilistic detection of short events, with application to critical care monitoring

**Norm Aleks**
U.C. Berkeley
*norm@cs.berkeley.edu*

**Stuart Russell**
U.C. Berkeley
*russell@cs.berkeley.edu*

**Michael G. Madden**
National U. of Ireland, Galway
*michael.madden@nuigalway.ie*

**Diane Morabito**
U.C. San Francisco
*morabitod@*
*neurosurg.ucsf.edu*

**Kristan Staudenmayer**
Stanford University
*kristans@*
*stanford.edu*

**Mitchell Cohen**
U.C. San Francisco
*mcohen@*
*sfghsurg.ucsf.edu*

**Geoffrey Manley**
U.C. San Francisco
*manleyg@*
*neurosurg.ucsf.edu*

## Abstract

We describe an application of probabilistic modeling and inference technology to the problem of analyzing sensor data in the setting of an intensive care unit (ICU). In particular, we consider the arterial-line blood pressure sensor, which is subject to frequent data artifacts that cause false alarms in the ICU and make the raw data almost useless for automated decision making. The problem is complicated by the fact that the sensor data are averaged over fixed intervals whereas the events causing data artifacts may occur at any time and often have durations significantly shorter than the data collection interval. We show that careful modeling of the sensor, combined with a general technique for detecting sub-interval events and estimating their duration, enables detection of artifacts and accurate estimation of the underlying blood pressure values. Our model's performance identifying artifacts is superior to two other classifiers' and about as good as a physician's.

## 1 Introduction

The work we report here falls under the general heading of *state estimation*, i.e., computing the posterior distribution $P(\mathbf{X_t}|\mathbf{e_{1:t}})$ for the state variables $\mathbf{X}$ of a partially observable stochastic system, given a sequence of observations $\mathbf{e_{1:t}}$. The specific setting for our work at the Center for Biomedical Informatics in Critical Care (C-BICC) is an intensive care unit (ICU) at San Francisco General Hospital (SFGH) specializing in traumatic brain injury, part of a major regional trauma center. In this setting, the state variables $\mathbf{X_t}$ include aspects of patient state, while the evidence variables $\mathbf{E_t}$ include up to 40 continuous streams of sensor data such as blood pressures (systolic/diastolic/mean, arterial and venous), oxygenation of blood, brain, and other tissues, intracranial pressure and temperature, inspired and expired oxygen and $CO_2$, and many other measurements from the mechanical ventilator.

A section of data from these sensors is shown in Figure 1(a). It illustrates a number of artifacts, including, in the top traces, sharp deviations in blood pressure due to external interventions in the arterial line; in the middle traces, ubiquitous drop-outs in the venous oxygen level; and in the lower traces, many jagged spikes in measured lung compliance due to coughing.

The artifacts cannot be modeled simply as "noise" in the sensor model; many are extended over time (some for as long as 45 minutes) and most exhibit complex patterns of their own. Simple techniques for "cleaning" such data, such as median filtering, fail. Instead, we follow the general approach suggested by Russell and Norvig (2003), which involves careful generative modeling of sensor state using dynamic Bayesian networks (DBNs).

This paper focuses on the arterial-line blood pressure sensor (Figure 1(b)), a key element of the monitoring system. As we describe in Section 2, this sensor is subject to multiple artifacts, including

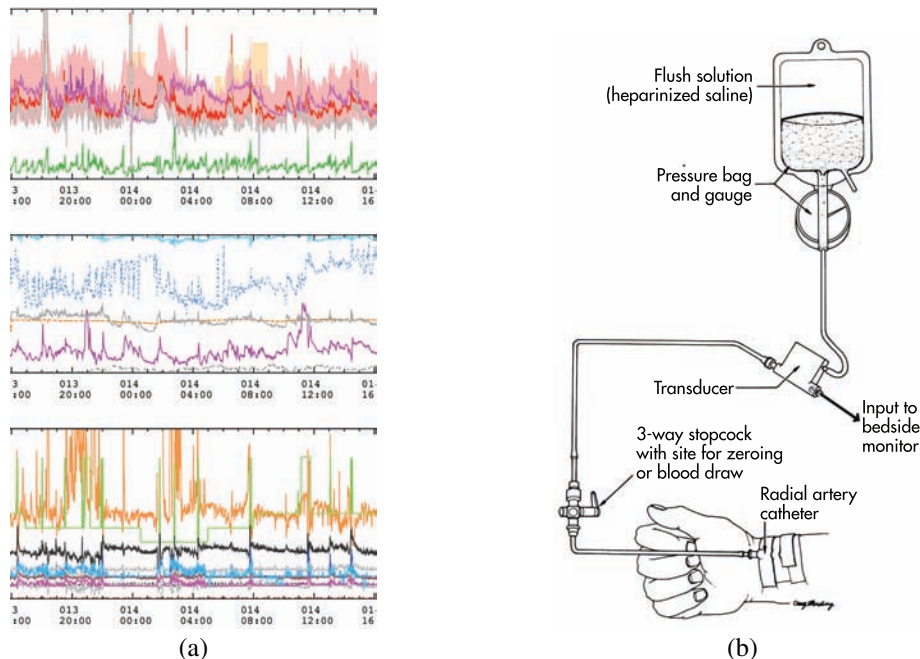

(a)                                    (b)

Figure 1: (a) One day's worth of minute-by-minute monitoring data for an ICU patient. (b) Arterial-line blood pressure measurement.

artificially low or high values due to zeroing, line flushes, or the drawing of blood samples. These artifacts not only complicate the state estimation and diagnosis task; they also corrupt recorded data and cause a large number of false alarms in the ICU, which lead in turn to true alarms being ignored and alarms being turned off (Tsien & Fackler, 1997). By modeling the artifact-generating processes, we hope to be able to infer the true underlying blood pressure even when artifacts occur.

To this point, the task described would be an applied Bayesian modeling problem of medium difficulty. What makes it slightly unusual and perhaps of more general interest is the fact that our sensor data are recorded as averages over each minute (our analysis is off-line, for the purpose of making recorded data useable for biomedical research), whereas the events of interest—in this case, re-zeroings, line flushes, and blood draws—can occur at any time and have durations ranging from under 5 seconds to over 100 seconds. Thus, the natural time step for modeling the sensor state transitions might be one second, whereas the measurement interval is much larger. This brings up the question of how a "slow" (one-minute) model might be constructed and how it relates to a "fast" (one-second) model. This is an instance of a very important issue studied in the dynamical systems and chemical kinetics literatures under the heading of *separation of time scales* (see, e.g., Rao & Arkin, 2003). Fortunately, in our case the problem has a simple, exact solution: Section 3 shows that a one-minute model can be derived efficiently, offline, from the more natural one-second model and gives exactly the same evidence likelihood. The more general problem of handling multiple time scales within DBNs, noted by Aliferis and Cooper (1996), remains open.

Section 4 describes the complete model for blood pressure estimation, including artifact models, and Section 5 then evaluates the model on real patient data. We show a number of examples of artifacts, their detection, and inference of the underlying state values. We analyze model performance over more than 300 hours of data from 7 patients, containing 228 artifacts. Our results show very high precision and recall rates for event detection; we are able to eliminate over 90% of false alarms for blood pressure while missing fewer than 1% of the true alarms.

Our work is not the first to consider the probabilistic analysis of intensive care data. Indeed, one of the best known early Bayes net applications was the ALARM model for patient monitoring under ventilation (Beinlich et al., 1989)—although this model had no temporal element. The work most closely related to ours is that of Williams, Quinn, and McIntosh (2005), who apply factorial switching Kalman filters—a particular class of DBNs—to artifact detection in neonatal ICU data. Their (one-second) model is roughly analogous to the models described by Russell and Norvig, using Boolean state variables to represent events that block normal sensor readings. Sieben and

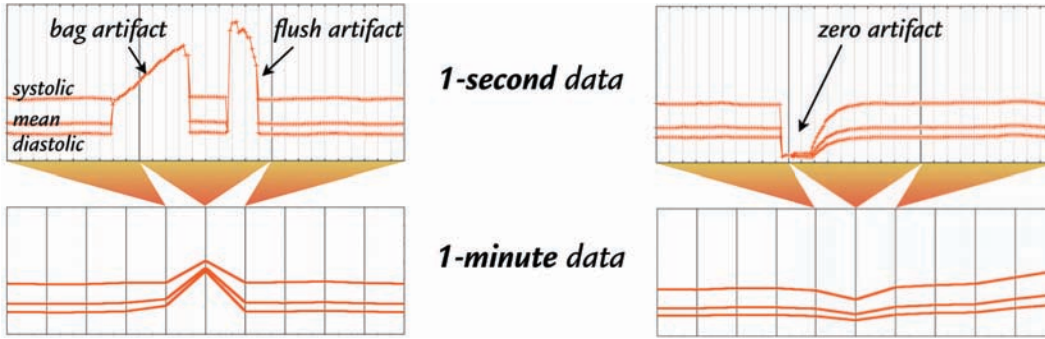

Figure 2: 1-second (top) and 1-minute-average (bottom) data for systolic/mean/diastolic pressures. One the left, a blood draw and line flush in quick succession. On the right, a zeroing.

Gather (2007) have applied discriminative models (decision forests and, more recently, SVMs) to correction of one-second-resolution heart-rate data. Another important line of work is the MIMIC project, which, like ours, aims to apply model-based methods to the interpretation of ICU data (Heldt et al., 2006).

## 2  Blood Pressure Monitoring

Blood pressure provides informs much of medical thinking and is typically measured continuously in the ICU. The most common ICU blood pressure measurement device is the *arterial line*, illustrated in Figure 1(b); a catheter placed into one of the patient's small arteries is connected to a pressure transducer whose output is displayed on a bedside monitor.

Because blood flow varies during the cardiac cycle, blood pressure is pulsatile. In medical records, including our data set, blood pressure measurements are summarized in two or three values: *systolic* blood pressure, which is the maximum reached during the cardiac cycle, *diastolic,* which is the corresponding minimum, and sometimes the *mean.*

We consider the three common artifact types illustrated in Figure 2: 1) in a *blood draw,* sensed pressure gradually climbs toward that of the pressure bag, then suddenly returns to the blood pressure when the stopcock is closed, seconds or minutes later; 2) in a *line flush,* the transducer is exposed to bag pressure for a few seconds; 3) during *zeroing*, the transducer is exposed to atmospheric pressure (defined as zero). We refer to blood draws and line flushes collectively as "bag events."

Figure 2(top) shows the artifacts using data collected at one-second intervals. However, the data we work with are the one-minute means of the one-second data, as shown in Figure 2(bottom). A fairly accurate simplification is that each second's reading reflects either the true blood pressure or an artifactual pressure, thus our model for the effect of averaging is that each recorded one-minute datum is a linear function of the true pressure, the artifactual pressure(s), and the fraction of the minute occupied by artifact(s). Using systolic pressure $s$ as an example, for an artifact of length $p$ (as a fraction of the averaging interval) and mean artifact pressure $x$, the apparent pressure $s = px + (1 \quad p)s$.

Our DBN model in Section 4 includes summary variables and equations relating the one-minute readings to the true underlying pressures, artifacts' durations, bag and atmospheric pressure, etc.; it can therefore estimate the duration and other characteristics of artifacts that have corrupted the data. Patterns produced by artifacts in the one-minute data are highly varied, but it turns out (see Section 5) that the detailed modeling pays off in revealing the characteristic relationships that follow from the nature of the corrupting events.

## 3  Modeling Sub-Interval Events

The data we work with are generated by a combination of physiological processes that vary over timescales of several minutes and artifactual events lasting perhaps only a few seconds. A natural

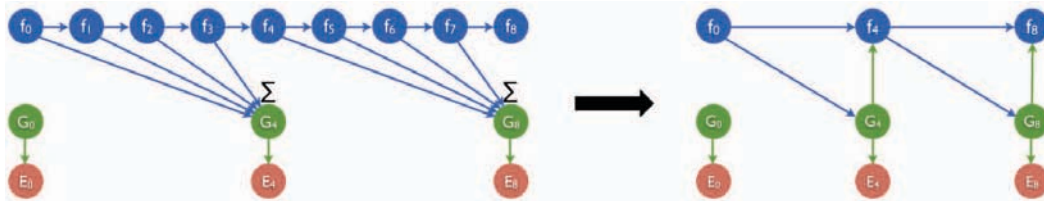

Figure 3: (left) DBN model showing relationships among the fast event variables $f_i$, interval count variables $G_{Nj}$, and measurement variables $E_{Nj}$. (right) A reduced model with the same distributions for $G_0, G_N, \ldots, G_{Nt}$.

choice would be a "fast" time step for the DBN model, e.g., 1 second: on this timescale, the sensor state variables indicate whether or not an artifactual event is currently in progress. The transition model for these variables indicates the probability at each second that a new event begins and the probability that an event already in progress continues. Assuming for now that there is only one event type, and given memoryless (geometric) distribution of durations such as we see in Section 5, only two parameters are necessary: $p = P(f_i = 1 \mid f_{i-1} = 1)$ and $q = P(f_i = 1 \mid f_{i-1} = 0)$. Both can be estimated simply by measuring event frequencies and durations.

The main drawback of using a fast time step is computational: inference must be carried out over 60 time steps for every one measurement that arrives. Furthermore, much of this inference is pointless given the lack of evidence at all the intervening time steps.

We could instead build a model with a "slow" time step of one minute, so that evidence arrives at each time step. The problem here is to determine the structure and parameters of such a model. First, to explain the evidence, we'll need a count variable saying how many seconds of the minute were occupied by events. It is easy to see that this variable must depend on the corresponding variable one minute earlier: for example, if the preceding minute was fully occupied by a blood draw event, then the blood draw was in progress at the beginning of the current minute, so the current minute is likely to be at least partially occupied by the event. (If there are multiple mutually exclusive event types, then each count variable depends on *all* the preceding variables.) Each count variable can take on 61 values, which leads to *huge* conditional distributions summarizing how the preceding 60 seconds could be divided among the various event types. Estimating these seems hopeless.

However, as we will now see, CPTs for the slow model need not be estimated or guessed—they can be *derived* from the fast model. This is the typical situation with separation of time scales: slow-time-scale models are computationally more tractable but can only be constructed by deriving them from a fast-time-scale model.

Consider a fast model as shown in Figure 3(a). Let the fast time step be and a measurement interval be $N$ (where $N = 60$ in our domain). $f_i = 1$ iff an event is occurring at time $i$ ; $G_{Nj} = \sum_{i=N(j-1)}^{Nj-1} f_i$ counts the number of fast time steps within the $j$th measurement interval during which an event is occurring. The $j$th observed measurement $E_{Nj}$ is determined entirely by $G_{Nj}$; therefore, it suffices to consider the joint distribution over $G_0, G_N, \ldots, G_{Nt}$.

To obtain a model containing only variables at the slow intervals, we simply need to sum out the $f_i$ variables other than the ones at interval boundaries. We can do this topologically by a series of arc reversal and node removal operations (Shachter, 1986); a simple proof by induction (omitted) shows that, regardless of the number of fast steps per slow step, we obtain the reduced structure in Figure 3(b). By construction, this model gives the same joint distribution for $G_0, G_N, \ldots, G_{Nt}$. Importantly, neither $f_{Nj}$ nor $G_{Nj}$ depends on $G_{N(j-1)}$.[1]

To complete the reduced model, we need the conditional distributions $P(G_{Nj} \mid f_{N(j-1)})$ and $P(f_{Nj} \mid f_{N(j-1)}G_{Nj})$. That is, how many "ones" do we expect in an interval, given the event status at the beginning of the interval, and what is the probability that an event is occurring at the beginning of the next interval, given also the number of ones in the current interval? Given the fast model's parameters $p$ and $q$, these quantities can be calculated offline using dynamic programming:

a table is constructed for the variables $f_i$ and $C_i$ for $i$ from 1 up to $N$, where $C_i$ is the number of ones up to $i-1$ and $C_0 = 0$. The recurrences for $f_i$ and $C_i$ are as follows:

$$P(C_i, f_i = 1 | f_0) = p\, P(C_{i-1} = C_i - 1, f_{i-1} = 1 | f_0) + q\, P(C_{i-1} = C_i, f_{i-1} = 0 | f_0) \tag{1}$$

$$P(C_i, f_i = 0 | f_0) = (1-p)\, P(C_{i-1} = C_i - 1, f_{i-1} = 1 | f_0) + (1-q)\, P(C_{i-1} = C_i, f_{i-1} = 0 | f_0) \tag{2}$$

Extracting the required conditional probabilities from the table is straightforward. The table is of size $O(N^2)$, so the total time to compute the table is negligible for any plausible value of $N$. Now we have the following result:

**Theorem 1** *Given the conditional distributions computed by Equations 1 and 2, the reduced model in Figure 3(b) yields the same distribution for the count sequence $G_0, G_N, \ldots, G_{Nt}$ as the fine-grained model in Figure 3(a).*

The conditional distributions that we obtain by dynamic programming have some interesting limit cases. In particular, when events are short compared to measurement intervals and occur frequently, we expect the dependence on $f_{N(j-1)}$ to disappear and the distribution for $G_{Nj}$ to be approximately Gaussian with mean $\frac{N}{1+p/(1-q)}$. When $p = q$, the $f_i$s become i.i.d. and $G_{Nj}$ is exactly binomial—the recurrences compute the binomial coefficients via Pascal's rule.

Generalizing the analysis to the case of multiple disjoint event types (i.e., $f_i$ takes on more than two values) is mathematically straightforward and the details are omitted. There is, however, a complexity problem as the number of event types increases. The count variables $G_{Nj}$, $H_{Nj}$, and so on at time $Nj$ are all dependent on each other given $f_{N(j-1)}$, and $f_{Nj}$ depends on all of them; thus, using the approach given above, the precomputed tables will scale exponentially with the number of event types. This is not a problem in our application, where we do not expect sensors to have more than a few distinct types of "error" state. Furthermore, if each event type occurs independently of the others (except for the mutual exclusion constraint), then the conditional distribution for the count variable of each depends not on the *combination* of counts for the other types but on the *sum* of those counts, leading to low-order polynomial growth in the table sizes.

The preceding analysis covers only the case in which $f_i$ depends just on $f_{i-1}$, leading to independently occurring events with a geometric length distribution. Constructing models with other length distributions is a well-studied problem in statistics and most cases can be well approximated with a modest increase in the size of the dynamic programming table. Handling non-independent event occurrence is often more important; for example, blood draws may occur in clusters if multiple samples are required. Such dependencies can be handled by augmenting the state with timer variables, again at modest cost.

Before we move on to describe the complete model, it is important to note that a model with a finer time scale that the measurement frequency *can* provide useful extra information. By analogy with sub-pixel localization in computer vision, such a model can estimate the *time of occurrence* of an event within a measurement interval.

## 4 Combined model

The complete model for blood pressure measurements is shown in Figure 4. It has the same basic structure as the reduced model in Figure 3(b) but extends it in various ways.

The evidence variables $\mathbf{E}_{\mathbf{Nj}}$ are just the three reported blood pressure values *ObservedDiaBP*, *ObservedSysBP*, and *ObservedMeanBP*. These reflect, with some Gaussian noise, idealized *Apparent___* values, determined in turn by

- the true time-averaged pressures: *TrueDiaBP*, *TrueSysBP*, and *TrueMeanBP*;

- the total duration of artifacts within the preceding minute (i.e., the $G_{Nj}$ variables): *BagTime* and *ZeroTime*;

- the average induced pressure to which the transducer is exposed during each event type: *BagPressure* and *ZeroPressure* (these have their own slowly varying dynamics).

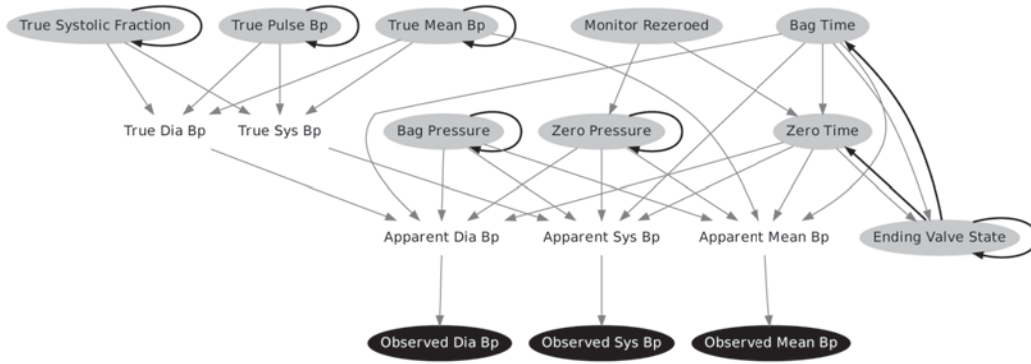

Figure 4: The blood pressure artifact detection DBN. Gray edges connect nodes within a time slice; black edges are between time slices. "Nodes" without surrounding ovals are deterministic functions included for clarity.

The *Apparent___* variables are deterministic functions of their parents. For example, we have

$$ApparentDiaBP \quad = \quad \frac{1}{N} \; BagTime \; BagPressure + ZeroTime \; ZeroPressure + $$
$$(N \quad BagTime \quad ZeroTime) \; TrueDiaBP$$

The physiological state variables in this model are *TrueSystolicFraction* (the average portion of each heartbeat spent ejecting blood), *TruePulseBP* (the peak-to-trough size of the pressure wave generated by each heartbeat), and *TrueMeanBP*. For simplicity, basic physiologic factors are modeled with random walks weighted toward physiologically sensible values.[2]

The key event variable in the model, corresponding to $f_{Nj}$ in Figure 3(b), is *EndingValveState*. This has three values for the three possible stopcock positions at the one-minute boundary: open to patient, open to bag, or open to air. The CPTs for this variable and for its children (at the next time step) *BagTime* and *ZeroTime* are the ones computed by the method of Section 3. The CPT for *EndingValveState* has 3　3　61　61 = 33　489 entries.

## 5 Experimental Results

To estimate the CPT parameters ($P(f_{t+1}=1 \; f_t=0)$ and $P(f_{t+1}=1 \; f_t=1)$) for the one-second model, and to evaluate the one-minute model's performance, we first needed ground truth for event occurrence and length. By special arrangement we were able to obtain 300 hours of 1Hz data, in which the artifacts we describe here are obvious to the human eye; one of us (a physician) then tagged each of those data points for artifact presence and type, giving the ground truth. (There were a total of 228 events of various lengths in the 300 hours' data.) With half the annotated data we verified that event durations were indeed approximately geometrically distributed, and estimated the one-second CPT parameters; from those, as described in Section 3, we calculated corresponding one-minute-interval CPTs.

Using averaging equivalent to that used by the regular system, we transformed the other half of the high-resolution data into 1-minute average blood pressures with associated artifact-time ground truth. We then used standard particle filtering (Gordon et al., 1993) with 8000 particles to derive posteriors for true blood pressure and the presence and length of each type of artifact at each minute. For comparison, we also evaluated three other artifact detectors:

a *support vector machine (SVM)* using blood pressures at times $t, t \quad 1, t \quad 2$, and $t \quad 3$ as its features;

a *deterministic model-based detector,* based on the linear-combination model of Section 2, which calculates three estimates of artifact pressure and length, pairwise among the current measured systolic, diastolic, and mean pressures, to explain the current measurements

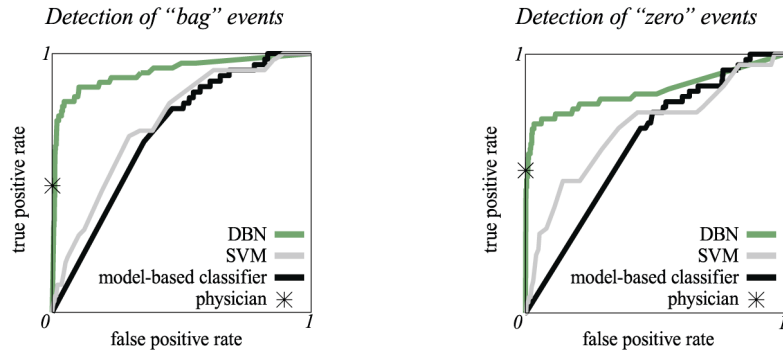

Figure 5: ROC curves for the DBN's performance detecting bag events (left) and zeroing events (right), as compared with an SVM, a deterministic model-based detector, and a physician.

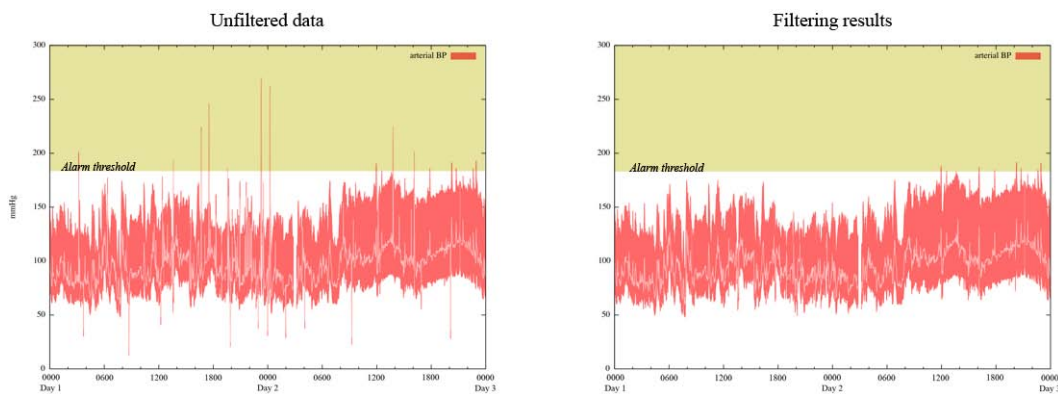

Figure 6: Two days' blood pressure data for one patient, with the hypertension threshold overlaid. Raw data are on the left; on the right are filtering results showing elimination (here) of false declarations of hypertension.

given the assumption that the true blood pressure is that recorded at the most recent minute during which no artifact was detected; it predicts artifact presence if the sum of the estimates' squared distances from their mean is below some threshold. (Because this model's prediction for any particular minute depends on its prediction at the previous minute, its sensitivity and specificity do not vary monotonically with changes in the threshold; the ROC curve shown is of only the undominated points.)

a *physician* working only with the one-minute-average data.

Figure 5(left) shows results for the detection of bag events. The DBN achieves a true positive rate of 80% with almost no false positives, or a TPR of 90% with 10% false positives. It does less well with zeroing events, as shown in Figure 5(b), achieving a TPR of nearly 70% with minimal false positives, but beyond that having unacceptable false positive levels. The physician had an even lower false positive rate for each artifact type, but with a true positive rate of only about 50%; the SVM and deterministic model-based detector both had better-than-chance performance but were clinically useless due to high false positive rates.

The model's accuracy in tracking true blood pressure is harder to evaluate because we have no minute-by-minute gold standard. (Arterial blood pressure measurements as we've described them, despite their artifacts, are the gold standard in the ICU. Placing a second arterial line, besides being subject to the same artifacts, also exposes patients to unnecessary infection risk.) However, on a more qualitative level, four physicians in our group have examined many hours of measured and inferred blood pressure traces, a typical example of which is shown in Figure 7, and have nearly always agreed with the inference results. Where the system's inferences are questionable, examining other sensors often helps to reveal whether a pressure change was real or artifactual.

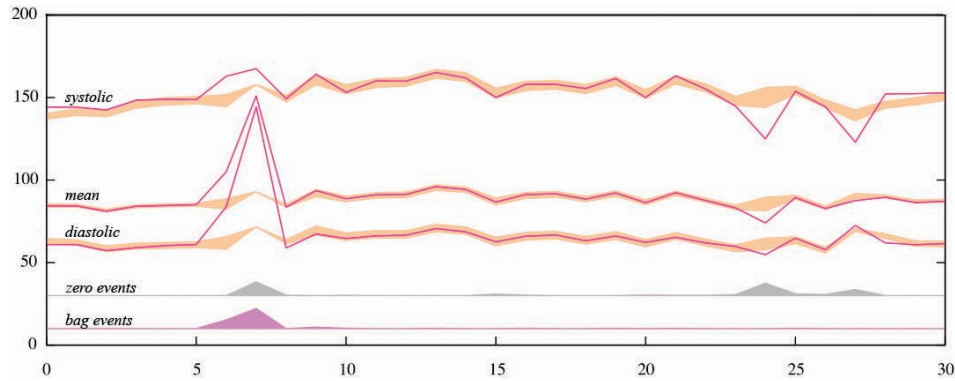

Figure 7: Sensed blood pressure (dark lines) and inferred true blood pressure (lighter bands, representing mean 1SD) across an observed blood draw with following zeroing. The lowest two lines show the inferred fraction of each minute occupied by bag or zero artifact.

## 6 Conclusions and Further Work

We have applied dynamic Bayesian network modeling to the problem of handling aggregated data with sub-interval artifacts. In preliminary experiments, this model of a typical blood pressure sensor appears quite successful at tracking true blood pressure and identifying and classifying artifacts.

Our approach has reduced the need for learning (as distinct from modeling and inference) to the small but crucial role of determining the distribution of event durations. It is interesting that the more straightforward learning approach, the SVM described above, had performance markedly inferior to the generative model's.

Modified to run at 1Hz, this model could run on-line at the bedside, helping to reduce false alarms. We are currently extending the model to include more sensors and physiological state variables and anticipate further improvements in detection accuracy as a result of combining multiple sensors.

## Footnotes

[1]Intuitively, the distribution over $G_{Nj}$ for the $N$th interval is determined by the value of $f$ at the beginning of the interval, independent of $G_{N(j-1)}$, whereas $f_{Nj}$ depends on the count $G_{Nj}$ for the preceding interval because, for example, a high count implies that an event is likely to be in progress at the end of the interval.

[2]More accurate modeling of the physiology actually improves the accuracy of artifact detection, but this point is explored in a separate paper.

## References

Aliferis, C., & Cooper, G. (1996). A structurally and temporally extended Bayesian belief network model: Definitions, properties, and modeling techniques. *Proc. Uncertainty in Artificial Intelligence* (pp. 28–39).

Beinlich, I., Suermondt, H., Chavez, R., & Cooper, G. (1989). The ALARM monitoring system. *Proc. Second European Conference on Artificial Intelligence in Medicine* (pp. 247–256).

Gordon, N. J., Salmond, D., & Smith, A. (1993). Novel approach to nonlinear/non-Gaussian Bayesian state estimation. *Radar and Signal Processing, IEE Proceedings–F*, *140*, 107–113.

Heldt, T., Long, W., Verghese, G., Szolovits, P., & Mark, R. (2006). Integrating data, models, and reasoning in critical care. *Proceedings of the 28th IEEE EMBS International Conference* (pp. 350–353).

Rao, C. V., & Arkin, A. P. (2003). Stochastic chemical kinetics and the quasi-steady-state assumption: Application to the Gillespie algorithm. *Journal of Chemical Physics*, *18*.

Russell, S. J., & Norvig, P. (2003). *Artificial intelligence: A modern approach*. Upper Saddle River, New Jersey: Prentice-Hall. 2nd edition.

Shachter, R. D. (1986). Evaluating influence diagrams. *Operations Research*, *34*, 871–882.

Sieben, W., & Gather, U. (2007). Classifying alarms in intensive care—analogy to hypothesis testing. *Lecture notes in computer science*, 130–138.

Tsien, C. L., & Fackler, J. C. (1997). Poor prognosis for existing monitors in the intensive care unit. *Critical Care Medicine*, *25*, 614–619.

Williams, C. K. I., Quinn, J., & McIntosh, N. (2005). Factorial switching Kalman filters for condition monitoring in neonatal intensive care. *NIPS*. Vancouver, Canada.

